# Rapid Visual Processing using Spike Asynchrony

**Simon J. Thorpe & Jacques Gautrais**
Centre de Recherche Cerveau & Cognition
F-31062 Toulouse
France
email thorpe@cerco.ups-tlse.fr

## Abstract

We have investigated the possibility that rapid processing in the visual system could be achieved by using the order of firing in different neurones as a code, rather than more conventional firing rate schemes. Using SPIKENET, a neural net simulator based on integrate-and-fire neurones and in which neurones in the input layer function as analog-to-delay converters, we have modeled the initial stages of visual processing. Initial results are extremely promising. Even with activity in retinal output cells limited to one spike per neuron per image (effectively ruling out any form of rate coding), sophisticated processing based on asynchronous activation was nonetheless possible.

## 1. INTRODUCTION

We recently demonstrated that the human visual system can process previously unseen natural images in under 150 ms (Thorpe et al, 1996). Such data, together with previous studies on processing speeds in the primate visual system (see Thorpe & Imbert, 1989) put severe constraints on models of visual processing. For example, temporal lobe neurones respond selectively to faces only 80-100 ms after stimulus onset (Oram & Perrett, 1992; Rolls & Tovee, 1994). To reach the temporal lobe in this time, information from the retina has to pass through roughly ten processing stages (see Fig. 1). If one takes into account the surprisingly slow conduction velocities of intracortical axons (< 1 ms-1, see Nowak & Bullier, 1997) it appears that the computation time within any cortical stage will be as little as 5-10 ms. Given that most cortical neurones will be firing below 100 spikes.s$^{-1}$, it is difficult to escape the conclusion that processing can be achieved with only one spike per neuron.

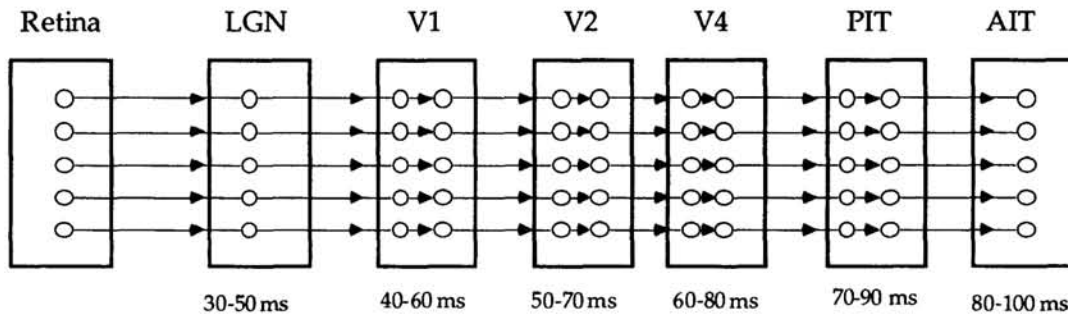

Figure 1 : Approximate latencies for neurones in different stages of the visual primate visual system (see Thorpe & Imbert, 1989; Nowak & Bullier, 1997).

Such constraints pose major problems for conventional firing rate codes since at least two spikes are needed to estimate a neuron's instantaneous firing rate. While it is possible to use the number of spikes generated by a population of cells to code analog values, this turns out to be expensive, since to code *n* analog values, one needs *n-1* neurones. Furthermore, the roughly Poisson nature of spike generation would also seriously limit the amount of information that can be transmitted. Even at 100 spikes.s$^{-1}$, there is a roughly 35% chance that the neuron will generate no spike at all within a particular 10 ms window, again forcing the system to use large numbers of redundant cells.

An alternative is to use information encoded in the temporal pattern of firing produced in response to transient stimuli (Mainen & Sejnowski, 1995). In particular, one can treat neurones not as *analog to frequency converters* (as is normally the case) but rather as *analog to delay converters*(Thorpe, 1990, 1994). The idea is very simple and uses the fact that the time taken for an integrate-and-fire neuron to reach threshold depends on input strength. Thus, in response to an intensity profile, the 6 neurones in figure 2 will tend to fire in a particular order - the most strongly activated cells firing first. Since each neuron fires one and only one spike, the firing rates of the cells contain no information, but there *is* information in the *order* in which the cells fire (see also Hopfield, 1995).

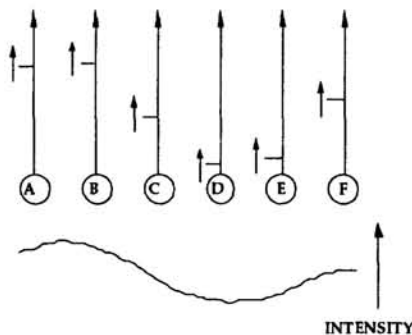

Figure 2 : An example of spike order coding. Because of the intrinsic properties of neurones the most strongly activated neurones will fire first. The sequence B>A>F>C>E>D is one ot the 720 (i.e. 6!) possible orders in which the 6 neurones can fire, each of which reflects a different intensity profile. Note that such a code can be used to send information very quickly.

To test the plausibility of using spike order rather than firing rate as a code, we have developed a neural network simulator "SPIKENET" and used it to model the initial stages of visual processing. Initial results are very encouraging and demonstrate that sophisticated visual processing can indeed be achieved in a visual system in which only one spike per neuron is available.

## 2. SPIKENET SIMULATIONS

SPIKENET has been developed in order to simulate the activity of large numbers of integrate-and-fire neurones. The basic neuronal elements are simple, and involve only a limited number of parameters, namely, an activation level, a threshold and a membrane time constant. The basic propagation mechanism involves processing the list of neurones that fired during the previous time step. For each spiking neuron, we add a synaptic weight value to each of its targets, and, if the target neuron's activation level exceeds its threshold, we add it to the list of spiking neurones for the next time step and reset its activation level by subtracting the threshold value. When a target neuron is affected for the first time on any particular time step, its activation level is recalculated to simulate an exponential decay over time. One of the great advantages of this kind of "event-driven" simulator is its computational efficiency - even very large networks of neurones can be simulated because no processor time is wasted on inactive neurones.

### 2.1 ARCHITECTURE

As an initial test of the possibility of single spike processing, we simulated the propagation of activity in a visual system architecture with three levels (see Figure 3). Starting from a gray-scale image (180 x 214 pixels) we calculate the levels of activation in two retinal maps, one corresponding to ON-center retinal ganglion cells, the other to OFF-center cells. This essentially corresponds to convolving the image with two Mexican-hat type operators. However, unlike more classic neural net models, these activation levels are not used to determine a continuous output value for each neuron, nor to calculate a firing rate. Instead, we treat the cells as analog-to-delay converters and calculate at which time step each retinal unit will fire. Because of their receptive field organization, cells which fire at the shortest latencies will correspond to regions in the image where the local contrast is high. Note, however, that each retinal ganglion cell will fire once and once only. While this is clearly not physiologically realistic (normally, cells firing at a short latencies go on to fire further spikes at short intervals) our aim was to see what sort of processing can be achieved in the absence of rate coding.

The ON- and OFF-center cells each make excitatory connections to a large number of cortical maps in the second level of the network. Each map contains neurones with a different pattern of afferent connections which results in orientation and spatial frequency selectivity similar to that described for simple-type neurones in striate cortex. In these simulations we used 32 different filters corresponding to 8 different orientations (each separated by 45°) and four different scales or spatial frequencies. This is functionally equivalent to having one single cortical map (equivalent to area V1) in which each point in visual space corresponds to a hypercolumn containing a complete set of orientation and spatial frequency tuned filters.

Units in the third layer receive weighted inputs from all the simple units corresponding to a particular region of space with the same orientation preference and thus roughly correspond to complex cells in area V1.

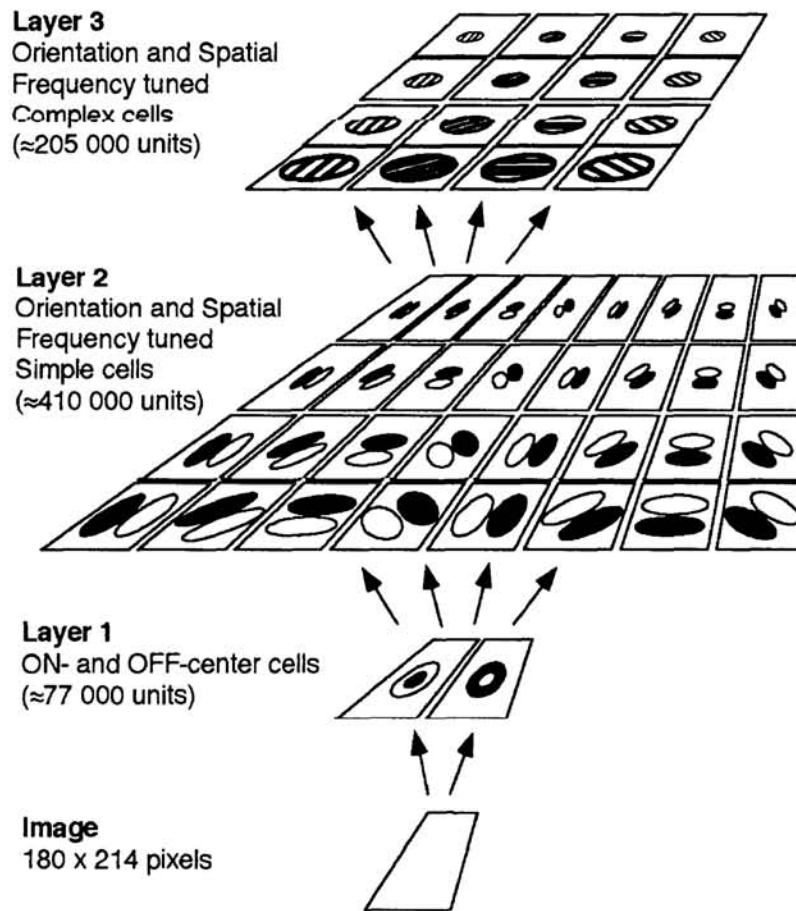

**Layer 3**
Orientation and Spatial
Frequency tuned
Complex cells
(≈205 000 units)

**Layer 2**
Orientation and Spatial
Frequency tuned
Simple cells
(≈410 000 units)

**Layer 1**
ON- and OFF-center cells
(≈77 000 units)

**Image**
180 x 214 pixels

Figure 3 : Architecture used in the present simulations

One unusual feature of the propagation process used in **SPIKENET** is that the post-synaptic effect of activating a synapse is not fixed, but depends on how many inputs have already been activated. Thus, the earliest firing cells produce a maximal post-synaptic effect (100%), but those which fire later produce less and less response. Specifically, the sensitivity of the post-synaptic neuron decreases by a fixed percentage each time one of its inputs fires. The phenomenon is somewhat similar to the sorts of activity-dependent synaptic depression described recently by Markram & Tsodyks (1996) and others, but differs in that the depression affects all the inputs to a particular neuron. The net result is to make the post-synaptic cell sensitive to the *order* in which its inputs are activated.

## 2.2 SIMULATION RESULTS

When a new image is presented to the network, spikes are generated asynchronously in the ON- and OFF-center cells of the retina in such a way that information about regions of the image with high local contrast (i.e. where there are contours present) are sent to the cortex first. Progressively, neurons in the second layer become more and more excited, and, after a variable number of time steps, the first cells in the second layer will start to

reach threshold and fire. Note that, as in the first layer, the earliest firing units will be those for whom the pattern of input activation best matches their receptive field structure.

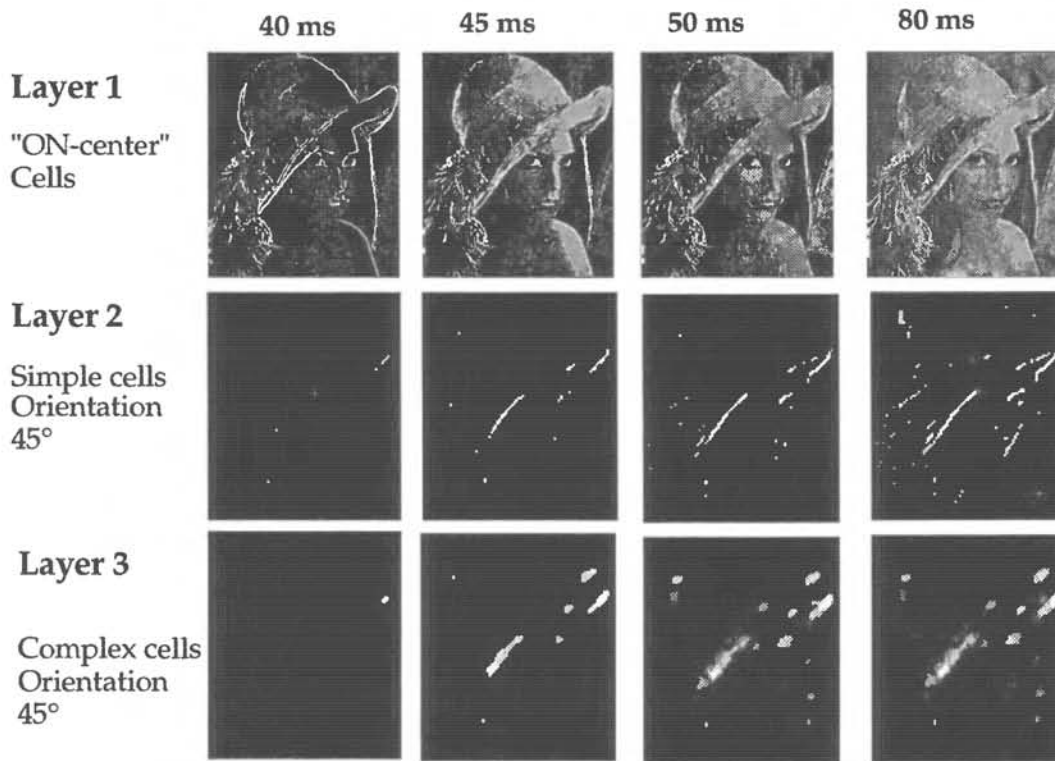

Figure 4 : Development of activity in 3 of the maps

Figure 4 illustrates this process for just three maps. The top row shows the location of units in the ON-center retinal map that have fired after various delays. After 40 msec, the main outlines of the figure can be seen but progressively more details are seen after 45 and then 50 ms. Note that the representation used here uses pixel intensity to code the order in which the cells have fired - bright white spots correspond to places in the image where the cells fired earliest. In the final frame (taken at 80 ms) the vast majority of ON-center cells have already fired and the resulting image is quite similar to a high spatial frequency filtered version of the original image.

The middle row of images shows activity in one of the second level maps - the one corresponding to medium spatial frequency components oriented at 45°. Note that in the first timeslice (40 ms) very few cells have fired, but that the proportion increases progressively over the next 10 or so milliseconds. However, even at the end of the propagation process, the proportion of cells that have actually fired remains low. Finally, the lowest row shows activity in the corresponding third layer map - again corresponding to contours oriented at 45°, but this time with less precise position specificity as a result of the grouping process.

Figure 5 plots the total number of spikes occurring per millisecond in each of the three layers during the first 100 ms following the onset of processing. It is perhaps not

surprising that the onset of firing occurs later for layers 2 and 3. However, there is a huge amount of overlap in the onset latencies of cells in the three layers, and indeed, it is doubtful whether there would be any systematic differences in mean onset latency between the three layers.

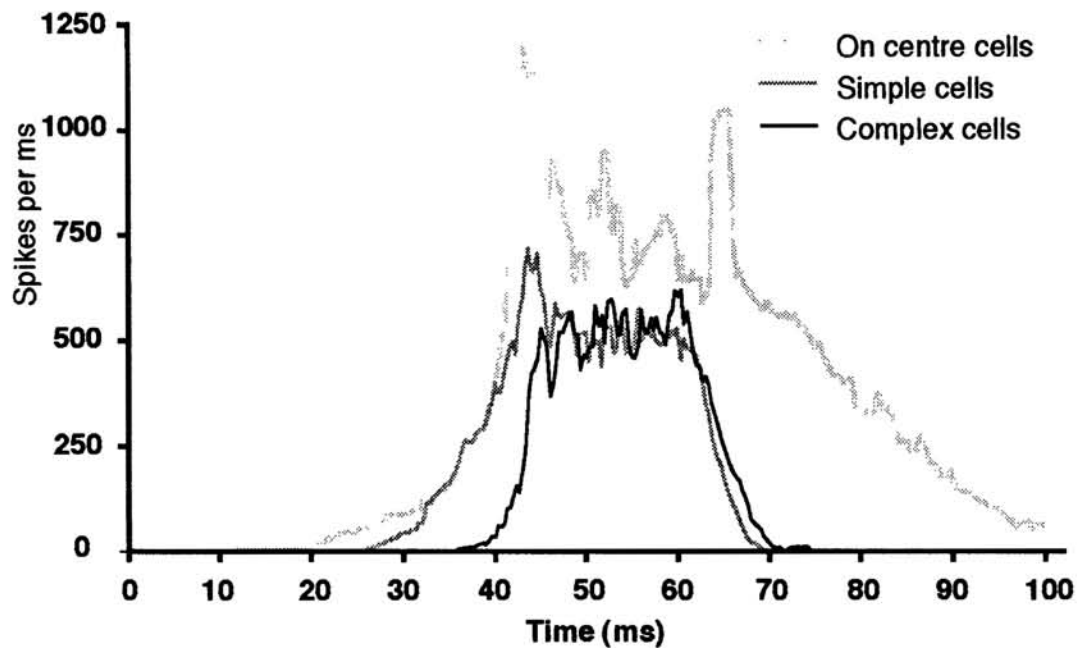

Figure 5 : Amount of activity measured in spikes/ms for the three layers of neurones as a function of time

But perhaps one of the most striking features of these simulations is the way in which the onset latency of cells can be seen to vary with the stimulus. The small number of cells in each layer which fire early are in fact very special because only the most optimally activated cells will fire at such short latencies. The implications of this effect for visual processing are far reaching because it means that the earliest information arriving at later processing stages will be particularly informative because the cells involved are very unambiguous. Interestingly, such changes in onset latency have been observed experimentally in neurones in area V1 of the awake primate in response to changes in orientation. In these experiments it was shown that shifting the orientation of a grating away from a neuron's preferred orientation could result in changes in not only the firing rate of the cell, but also increases in onset latency of as much as 20-30 ms (Celebrini, Thorpe, Trotter & Imbert, 1993).

## 3.  CONCLUSIONS

A number of points can be made on the basis of these results. Perhaps the most important is that visual processing can indeed be performed under conditions in which spike frequency coding is effectively ruled out. Clearly, under normal conditions, neurones in the visual system that respond to a visual input will almost invariably generate more

than one spike. However, as we have argued previously, processing in the visual system is so fast that most cells will not have time to generate more than one spike before processing in later stages has to be initiated. The present results indicate that the use of temporal order coding may provide a key to understanding this remarkable efficiency.

The simulations presented here are clearly very limited, but we are currently looking at spike propagation in more complex architectures that include extensive horizontal connections between neurones in a particular layer as well as additional layers of processing. As an example, we have recently developed an application capable of digit recognition. SPIKENET is well suited for such large scale simulations because of the event-driven nature of the propagation process. For instance, the propagation presented here, which involved roughly 700 000 neurones and over 35 million connections, took roughly 15 seconds on a 150 MHz PowerMac, and even faster simulations are possible using parallel processing. With this is view we have developed a version of SPIKENET that uses PVM (Parallel Virtual Machine) to run on a cluster of workstations.

## References

Celebrini S., Thorpe S., Trotter Y. & Imbert M. (1993). Dynamics of orientation coding in area V1 of the awake primate *Visual Neuroscience* **10**, 811-25.

Hopfield J. J. (1995). Pattern recognition computation using action potential timing for stimulus representation. *Nature*, **376**, 33-36.

Mainen Z. F. & Sejnowski T. J. (1995). Reliability of spike timing in neocortical neurons *Science*, **268**, 1503-6.

Markram H. & Tsodyks M. (1996) Redistribution of synaptic efficacy between neocortical pyramidal neurons. *Nature*, **382**, 807-810

Nowak L.G. & Bullier J (1997) The timing of information transfer in the visual system. In Kaas J., Rocklund K. & Peters A. (eds). Extrastriate Cortex in Primates (in press). Plenum Press.

Oram M. W. & Perrett D. I. (1992). Time course of neural responses discriminating different views of the face and head *Journal of Neurophysiology*, **68**, 70-84.

Rolls E. T. & Tovee M. J. (1994). Processing speed in the cerebral cortex and the neurophysiology of visual masking *Proc R Soc Lond B Biol Sci*, **257**, 9-15.

Thorpe S., Fize D. & Marlot C. (1996). Speed of processing in the human visual system *Nature*, **381**, 520-522.

Thorpe S. J. (1990). Spike arrival times: A highly efficient coding scheme for neural networks. In R. Eckmiller, G. Hartman & G. Hauske (Eds.), *Parallel processing in neural systems* (pp. 91-94). North-Holland: Elsevier. Reprinted in H. Gutfreund & G. Toulouse (1994), *Biology and computation : A physicist's choice.* Singapour: World Scientific.

Thorpe S. J. & Imbert M. (1989). Biological constraints on connectionist models. In R. Pfeifer, Z. Schreter, F. Fogelman-Soulié & L. Steels (Eds.), *Connectionism in Perspective.* (pp. 63-92). Amsterdam: Elsevier.
